# Divisive and Subtractive Mask Effects: Linking Psychophysics and Biophysics

**Barbara Zenger**
Division of Biology
Caltech 139-74
Pasadena, CA 91125
*barbara@klab.caltech.edu*

**Christof Koch**
Computation and Neural Systems
Caltech 139-74
Pasadena, CA 91125
*koch@klab.caltech.edu*

## Abstract

We describe an analogy between psychophysically measured effects in contrast masking, and the behavior of a simple integrate-and-fire neuron that receives time-modulated inhibition. In the psychophysical experiments, we tested observers ability to discriminate contrasts of peripheral Gabor patches in the presence of collinear Gabor flankers. The data reveal a complex interaction pattern that we account for by assuming that flankers provide divisive inhibition to the target unit for low target contrasts, but provide subtractive inhibition to the target unit for higher target contrasts. A similar switch from divisive to subtractive inhibition is observed in an integrate-and-fire unit that receives inhibition modulated in time such that the cell spends part of the time in a high-inhibition state and part of the time in a low-inhibition state. The similarity between the effects suggests that one may cause the other. The biophysical model makes testable predictions for physiological single-cell recordings.

## 1 Psychophysics

Visual images of Gabor patches are thought to excite a small and specific subset of neurons in the primary visual cortex and beyond. By measuring psychophysically in humans the contrast detection and discrimination thresholds of peripheral Gabor patches, one can estimate the sensitivity of this subset of neurons. Furthermore, spatial interactions between different neuronal populations can be probed by testing the effects of additional Gabor patches (masks) on performance. Such experiments have revealed a highly configuration-specific pattern of excitatory and inhibitory spatial interactions [1, 2].

### 1.1 Methods

Two vertical Gabor patches with a spatial frequency of 4cyc/deg were presented at 4 deg eccentricity left and right of fixation, and observers had to report which patch had the higher contrast (spatial 2AFC). In the "flanker condition" (see Fig. 1A),

the two targets were each flanked by two collinear Gabor patches of 40% contrast, presented above and below the targets (at a distance of 0.75 deg, i.e., 3 times the spatial period of the Gabor).

Observers fixated a central cross, which was visible before and during each trial, and then initiated the trial by pressing the space bar on the computer keyboard. Two circular cues appeared for 180ms to indicate the locations of the two targets (to minimize spatial uncertainty). A blank stimulus of randomized length (500ms±100ms) was followed by a 83ms stimulus presentation. No mask was presented. Observers indicated which target had the higher contrast ("left" or "right") by specified keys. Auditory feedback was provided.

Thresholds were determined using a staircase procedure [3]. Whenever the staircase procedure showed a ceiling effect (asking to display contrasts above 100%) the data for this pedestal contrast in this condition were not considered for this observer, even if at other days valid threshold estimates were obtained, because considering only the 'good days' would have introduced a bias. Seven observers with normal or corrected-to-normal vision participated in the experiment. Each condition was repeated at least six times. The experimental procedure is in accordance with Caltech's Committee for the Protection of Human Subjects.

Experiments were controlled by an O2 Silicon Graphics workstation, and stimuli were displayed on a raster monitor. Mean luminance $L_m$ was set to 40 cd/m$^2$. We used color-bit stealing to increase the number of grey levels that can be displayed [4]. A gamma correction ensured linearity of the gray levels.

To remove some of the effects of inter-observer variability from our data analysis, the entire data set of each observer was first normalized by his or her average performance across all conditions, and only then averages and standard errors were computed. The mean standard errors across all conditions and contrast levels are presented as bars in Figs. 1B and 1D.

## 1.2 Results

In the absence of flankers (circles, Fig. 1B), discrimination thresholds first decrease from absolute detection threshold at 8.7% with increasing pedestal contrast and then increase again. As common in sensory psychophysics, we assume that the contrast discrimination thresholds can be derived from an underlying sigmoidal contrast-response function $r(c)$ (see Fig. 1C, solid curve), together with the assumption that some fixed response difference $\Delta r = 1$ is required for correct discrimination [2]. In other words, for any fixed pedestal contrast $c$, the discrimination threshold $\Delta c$ satisfies $r(c + \Delta c) = r(c) + 1$.

Our underlying assumption is that at the decision stage, the level of noise in the signal is independent of the response magnitude. Neuronal noise, on the other hand, is usually well characterized by a Poisson process, that is, the noise level increases with increasing response. Little evidence exists, however, that this "early" response dependent noise actually limits the performance. It is conceivable that this early noise is relatively small, that the performance-limiting noise is added at a later processing stage, and that this noise is independent of the response magnitude.

To describe the response $r$ of the system to a single, well-isolated, target as a function of its contrast $c$ we adopt the function suggested by Foley (1994) [2]:

$$r_{\text{isolated}}(c) = \frac{ac^p}{c^{p-q} + c_{\text{th}}^{p-q}}. \tag{1}$$

For plausible parameters ($c, c_{\text{th}} > 0$) this function is proportional to $c^p$ for $c \ll c_{\text{th}}$

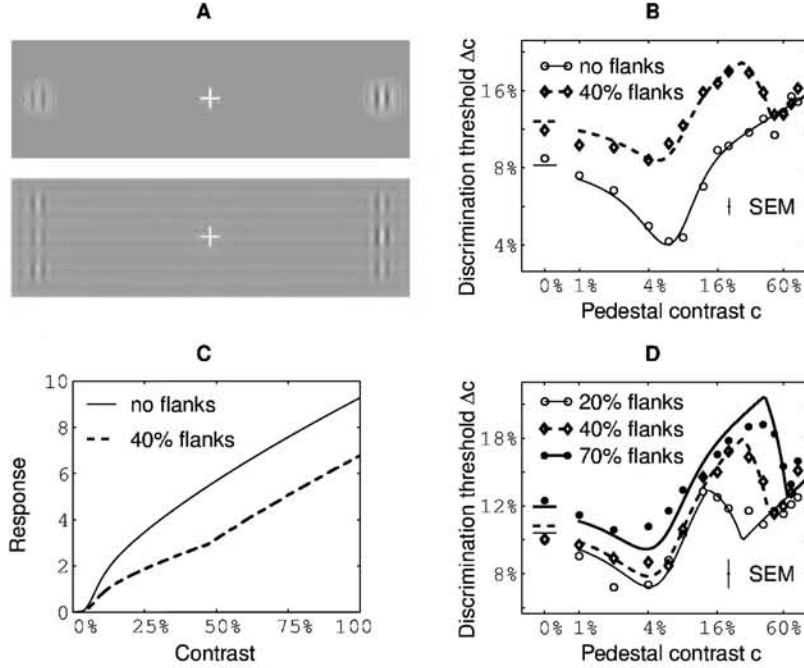

Figure 1: (A) Sample stimuli without flanks and with flanks. (B) Discrimination thresholds average across seven observers for flanked (diamond) and unflanked (circles) targets. (C) Contrast response functions used for model prediction in (B). (D) Discrimination performance averaged across four observers for different flank contrasts. Lines in (B) and (D) represent the best model fit.

and is proportional to $c^q$ for $c \gg c_{\text{th}}$, consistent with a modified Weber law [5]. The contrast-response function obtained for the parameters given in the first row of Tab. 1 is shown in Fig. 1C (solid line). The corresponding discrimination thresholds (Fig. 1B; solid line) fit well the psychophysical data (open circles).

What happens to the dipper function when the two targets are flanked by Gabor patches of 40% contrast? In the presence of flankers, contrast discrimination thresholds (diamonds, Fig. 1B) first decrease, then increase, then decrease again, and finally increase again, following a W-shaped function. Depending on target contrast, one can distinguish two distinctive flanker effects: for targets of 40% contrast or less, flankers impair discrimination. In the masking literature such suppressive effects are often attributed to a divisive input from the mask to the target; in other words, the flanks seem to reduce the target's gain [2]. For targets of 50% or more (four rightmost data points in Fig. 1B), contrast performance is about the same irrespective of whether flankers are present or not; at these high target contrasts, flankers apparently cease to contribute to the target's gain control.

Following this concept, we define two model parameters to describe the effects of the flankers: the first parameter, $c_0$, determines the maximal target contrast at which gain control is still effective; the second parameter, $b$, determines the strength of

the gain control. Formally written, we obtain:

$$r_{\text{flanked}}(c) = \begin{cases} r_{\text{isolated}}(c)/b & \text{for } c \leq c_{\text{o}}, \text{ (gain control)} \\ r_{\text{isolated}}(c) - d & \text{for } c \geq c_{\text{o}}, \text{ (no gain control)} \end{cases} \quad (2)$$

In the low-contrast range, the contrast-response functions with and without flankers are multiples of each other (factor $b$); in the high-contrast regime, the two curves are shifted vertically (offset $d$) with respect to each other (see Fig. 1C). The subtractive constant $d$ is not a free parameter, but is determined by imposing that $r$ be continuous at $c = c_{\text{o}}$, i.e., $r_{\text{isolated}}(c_{\text{o}})/b = r_{\text{isolated}}(c_{\text{o}}) - d$.

The parameters that best account for the average data in Figs. 1B and 1D in the least-mean-square sense were estimated using a multidimensional simplex algorithm [6].

Table 1: Best fitting model parameters in the least-square sense.

|  |  | no flanks |  |  | 20% flanks |  | 40% flanks |  | 70% flanks |  |
|---|---|---|---|---|---|---|---|---|---|---|
|  | $a$ | $c_{\text{th}}$ | $p$ | $q$ | $b$ | $c_{\text{o}}$ | $b$ | $c_{\text{o}}$ | $b$ | $c_{\text{o}}$ |
| Fig. 1BC | 0.363 | 7.14% | 4.47 | 0.704 | — | — | 1.86 | 46.8% | — | — |
| Fig. 1D | 0.395 | 6.07% | 3.78 | 0.704 | 1.69 | 26.4% | 1.78 | 44.9% | 2.01 | 64.3% |

Increasing the flanker contrast leads both to an increase in the strength of gain control $b$ and to an increase in the range $c_{\text{o}}$ in which gain control is effective. The predicted discrimination performance is shown superimposed on the data in Fig. 1B and D. As one can see, the model captures the behavior of the data reasonably well, considering that for each combined fit there are only four parameters to fit the unflanked data and two additional parameters for each W curve. Or, put differently, we use but two degrees of freedom to go from the unflanked to the flanked conditions.

## 2   Biophysics

While the above model explains the data, it remains a puzzle how the switch from divisive to subtractive is implemented neuronally. Here, we show that time-modulated inhibition can naturally account for the observed switch, without assuming input-dependent changes in the network.

### 2.1   Circuit Model

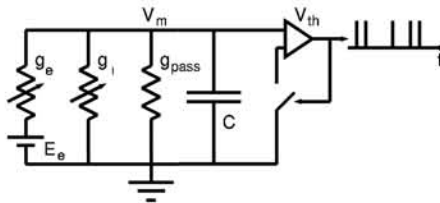

Figure 2: Circuit model used for the simulations.

To simulate the behavior of individual neurons we use a variant of the leaky integrate-and-fire unit (battery $E_{\text{e}} = 70$mV, capacitance $C = 200$pF, leak conductance $g_{\text{pass}} = 10$nS, and firing threshold $V_{\text{th}} = 20$mV, see Fig. 2).

Excitatory and inhibitory synaptic input are modeled as changes in the conductances $g_e$ and $g_i$, respectively. Whenever the membrane potential $V_m$ exceeds threshold ($V_{th}$), a spike is initiated and the membrane potential $V_m$ is reset to $V_{rest} = 0$. No refractory period was assumed. The model was implemented on a PC using the programming language C.

## 2.2 Simulations

Firing rates for increasing excitation ($g_e$) at various levels of inhibtion ($g_i$) are shown in Fig. 3A. For low excitatory input the cell never fires, because the input current is counter-balanced by the leakage current, thus preventing the cell from reaching its firing threshold. Once the cell does start firing, firing rates first increase very fast, but then rapidly converge against a linear function, whose slope is independent of $g_i$. When the inhibitory input is modulated in time and switches between a low

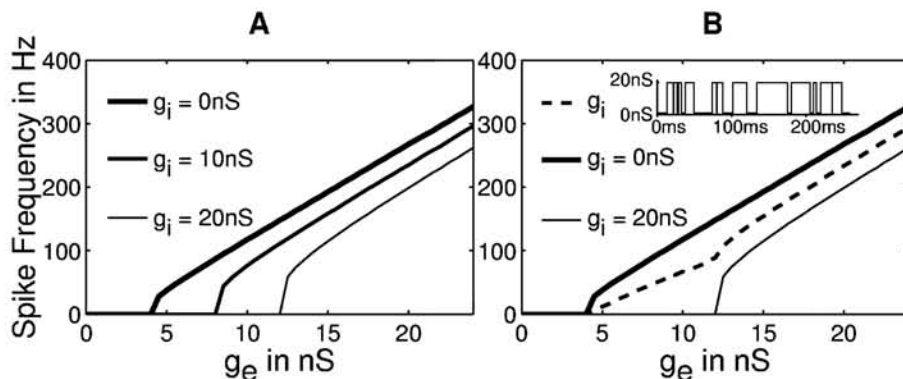

Figure 3: Simulations of circuit model with constant inhibition (A) or time-mdoulated inhbition (dashed line in (B)). This simple single-cell model matches the psychophysics remarkably well.

inhibition state ($g_i = g_{low}$) and a high inhibition state ($g_i = g_{high}$), the results look different (Fig. 3B, dashed line). The cell fires part of the time like a lowly inhibited cell, part of the time like a highly inhibited cell, explaining why the overall firing rate resemble weighted averages of the curves for constant $g_i$. A comparison of the no-inhibition curve ($g_i = 0$) and the curve for time-modulated inhbition demonstrates that inhibition switches from a divisive mode to a subtractive mode for increasing $g_e$. The $g_e$-level at which the switch occurs depends on the level of inhibition in the high-inhibition state (here $g_{high}$=20nS). The strength of divisive inhibition depends on the percentage of time $R$ that the cell spends in the high-inhibition state; in the example shown as a dashed line in Fig. 2B, the cell spends on average half of the time in the high-inhibition stage (thus $R$=50%), and remains the rest of the time in the low-inhibition stage.

## 3  Discussion

Both the psychophysical data and the biophysical model show a switch from divisive to subtractive inhibition. Making the connection between psychophysics and

biophysics explicitly, requires that a number of assumptions be made: (1) the excitatory input $g_e$ to the target unit increases with increasing target contrast; (2) increasing the flank contrast leads to an increase of $g_{high}$ (to account for the fact that the transition from divisive to subtractive inhibition occurs at higher contrasts $c_o$); (3) the relative time spent in the $g_{high}$ state ($R$) increases with flanker contrast (leading to a stronger divisive inhibition $b$ that is reflected in the overall performance decrease with increasing flanker contrast).

While these assumptions all seem quite plausible, there remains the question of why one would assume time-modulated inhbition in the first place. Here we suggest three different mechanisms: First, the time-modulation might reflect inhibitory input by synchronized interneurons [7], i.e., sometimes a large number of them fire at the same time (high-inhibition state) while at other times almost none of the inhibitory cells fire (low-inhibition state).

A second plausible implementation (which gives very similar results) assumes that there is only one transition and that the low- and high-inhibition state follow each other sequentially (rather than flipping back and forth as suggested in Fig. 3B). Indeed, cells in primary visual cortex often show a transient response at stimulus onset (which may reflect the low-inhibition state), followed by a smaller level of sustained response (which may reflect the high-inhibition state). In this context, $R$ would simply reflect the time delay between the onset of excitation and inhibition (with a large $R$ representing brief delays before inhibition sets in).

Finally, low- and high- inhibition states may reflect different subtypes of neurons which receive different amount of surround inhibition. In other words, some neurons are strongly inhibited (high-inhibition state) while others are not (low-inhibition state). The ratio of strongly-inhibited units (among all units) is given by $R$. The mean response of all the neurons will show a divisive inhibition in the range where the inhibited neurons are shut off completely, but will show a subtractive inhibition as soon as the inhibited units start firing.

To summarize on a more abstract level: any mechanism that will average firing rates of different $g_i$ states, rather than averaging different inhibitory inputs $g_i$, will lead to a mechanism that shows this switch from divisive to subtractive inhibition.

The remaining differences between the psychophysically estimated contrast-response functions (Fig. 1C) and the firing rates of the circuit model (Fig. 3B) seem to reflect mainly oversimplifications in the biophysical model. Saturation at high $g_e$ values, for instance, could be achieved by assuming refractory periods or other firing-rate adaptation mechanisms. The very steep slope directly after the switch from divisive to subtractive inhibition would disappear if the simple integrate-and-fire unit would be replaced by a more realistic unit in which — due to stochastic linearization — the firing rate rises more gradually once the threshold is crossed. In any case, one does not expect a precise match between the two functions, as psychophysical performance presumably relies on a variety of different neurons with different dynamic ranges. Once the model includes many neurons, one would need to define decision strategies. We believe that such a link between a biophysical model and psychophysical data is in principle possible, but have favored here simplicity at the expense of achieving a more quantitative match.

Our analysis of the circuit model shows that the psychophysical data can be explained without assuming complex interaction patterns between different neuronal units. While we have no reason to believe that the switching-mechanism from divisive to subtractive inhibition will become ineffective when considering large number of neurons, it does not require a large network. Our model suggests that the critical events happen at the level of individual neurons, and not in the network.

Our model makes two clear predictions: first, the contrast-response function of single neurons should show — in the presence of flankers — a switch from divisive to subtractive inhibition (Fig. 1C and Fig. 3B). Physiological studies have measured how stimuli outside the classical receptive field affect the absolute response level of the target unit [8, 9]. Distinguishing subtractive and divisive inhibition, however, requires that, in addition, surround effects on the *slope* of the contrast-response functions are estimated. Such experiments have been carried out by Sengpiel et al [10] in cat primary visual cortex. Their extracellular recordings show that when a target grating is surrounded by a high-contrast annulus, inhibition is indeed well described by a divisive effect on the response. It remains to be seen, however, whether surround annuli whose contrast is lower than the target contrast will act subtractively. The second prediction is that inhibition is bistable, *i.e.*, that there are distinct low- and high-inhibition states. These states may alternate in time within the same neuron, or they may be represented by different subsets of neurons.

## Acknowledgments

We would like to thank Jochen Braun, Gary Holt, and Laurent Itti for helpful comments. The research was supported by NSF, NIMH and a Caltech Divisional Scholarship to BZ.

## References

[1] U. Polat and D. Sagi. Lateral interactions between spatial channels: Suppression and facilitation revealed by lateral masking experiments. *Vision Research*, 33:993–999, 1993.

[2] John M. Foley. Human luminance pattern-vision mechanisms: masking experiments require a new model. *Journal of the Optical Society of America A*, 11:1710–1719, 1994.

[3] H. Levitt. Transformed up-down methods in psychoacoustics. *The Journal of the Acoustical Society of America*, 49:467–477, 1971.

[4] C.W. Tyler. Colour bit-stealing to enhance the luminance resolution of digital displays on a single pixel basis. *Spatial Vision*, 10(4):369–377, 1997.

[5] Gordon E. Legge. A power law for contrast discrimination. *Vision Research*, 21:457–467, 1981.

[6] W.H. Press, S.A. Teukolsky, W.T. Vetterling, and B.P. Flannery. *Numerical Recipes in C*. Cambridge University Press, 1992.

[7] W. Singer and C.M. Gray. Visual feature integration and the temporal correlation hypothesis. *Annual review of neuroscience*, 18:555–586, 1995.

[8] J.B. Levitt and J.S. Lund. Contrast dependence of contextual effects in primate visual cortex. *Nature*, 387:73–76, 1997.

[9] U. Polat, K. Mizobe, M.W. Pettet, T. Kasamatsu, and A.M. Norcia. Collinear stimuli regulate visual responses depending on cell's contrast threshold. *Nature*, 391:580–584, 1998.

[10] F. Sengpiel, R.J. Baddeley, T.C.B Freeman, R. Harrad, and C. Blakemore. Different mechanisms underlie three inhibitory phenomena in cat area 17. *Vision Research*, 38(14):2067–2080, 1998.
